# Bayesian Kernel Shaping for Learning Control

**Jo-Anne Ting**[1]**, Mrinal Kalakrishnan**[1]**, Sethu Vijayakumar**[2] **and Stefan Schaal**[1,3]

[1]Computer Science, U. of Southern California, Los Angeles, CA 90089, USA
[2]School of Informatics, University of Edinburgh, Edinburgh, EH9 3JZ, UK
[3]ATR Computational Neuroscience Labs, Kyoto 619-02, Japan

## Abstract

In kernel-based regression learning, optimizing each kernel individually is useful when the data density, curvature of regression surfaces (or decision boundaries) or magnitude of output noise varies spatially. Previous work has suggested gradient descent techniques or complex statistical hypothesis methods for local kernel shaping, typically requiring some amount of manual tuning of meta parameters. We introduce a Bayesian formulation of nonparametric regression that, with the help of variational approximations, results in an EM-like algorithm for simultaneous estimation of regression and kernel parameters. The algorithm is computationally efficient, requires no sampling, automatically rejects outliers and has only one prior to be specified. It can be used for nonparametric regression with local polynomials or as a novel method to achieve nonstationary regression with Gaussian processes. Our methods are particularly useful for learning control, where reliable estimation of local tangent planes is essential for adaptive controllers and reinforcement learning. We evaluate our methods on several synthetic data sets and on an actual robot which learns a task-level control law.

## 1 Introduction

Kernel-based methods have been highly popular in statistical learning, starting with Parzen windows, kernel regression, locally weighted regression and radial basis function networks, and leading to newer formulations such as Reproducing Kernel Hilbert Spaces, Support Vector Machines, and Gaussian process regression [1]. Most algorithms start with parameterizations that are the same for all kernels, independent of where in data space the kernel is used, but later recognize the advantage of locally adaptive kernels [2, 3, 4]. Such locally adaptive kernels are useful in scenarios where the data characteristics vary greatly in different parts of the workspace (e.g., in terms of data density, curvature and output noise). For instance, in Gaussian process (GP) regression, using a nonstationary covariance function, e.g., [5], allows for such a treatment. Performing optimizations individually for every kernel, however, becomes rather complex and is prone to overfitting due to a flood of open parameters. Previous work has suggested gradient descent techniques with cross-validation methods or involved statistical hypothesis testing for optimizing the shape and size of a kernel in a learning system [6, 7].

In this paper, we consider local kernel shaping by averaging over data samples with the help of locally polynomial models and formulate this approach, in a Bayesian framework, for both function approximation with piecewise linear models and nonstationary GP regression. Our local kernel shaping algorithm is computationally efficient (capable of handling large data sets), can deal with functions of strongly varying curvature, data density and output noise, and even rejects outliers automatically. Our approach to nonstationary GP regression differs from previous work by avoiding Markov Chain Monte Carlo (MCMC) sampling [8, 9] and by exploiting the full nonparametric characteristics of GPs in order to accommodate nonstationary data.

One of the core application domains for our work is learning control, where computationally efficient function approximation and highly accurate local linearizations from data are crucial for deriving controllers and for optimizing control along trajectories [10]. The high variations from fitting noise, seen in Fig. 3, are harmful to the learning system, potentially causing the controller to be unstable. Our final evaluations illustrate such a scenario by learning an inverse kinematics model for a real robot arm.

## 2  Bayesian Local Kernel Shaping

We develop our approach in the context of nonparametric locally weighted regression with locally linear polynomials [11], assuming, for notational simplicity, only a one-dimensional output—extensions to multi-output settings are straightforward. We assume a training set of $N$ samples, $D = \{\mathbf{x}_i, y_i\}_{i=1}^N$, drawn from a nonlinear function $y = f(\mathbf{x}) + \epsilon$ that is contaminated with mean-zero (but potentially heteroscedastic) noise $\epsilon$. Each data sample consists of a $d$-dimensional input vector $\mathbf{x}_i$ and an output $y_i$. We wish to approximate a locally linear model of this function at a query point $\mathbf{x}_q \in \Re^{d \times 1}$ in order to make a prediction $y_q = \mathbf{b}^T \mathbf{x}^q$, where $\mathbf{b} \in \Re^{d \times 1}$. We assume the existence of a spatially localized weighting kernel $w_i = K(\mathbf{x}_i, \mathbf{x}_q, \mathbf{h})$ that assigns a weight to every $\{\mathbf{x}_i, y_i\}$ according to its Euclidean distance in input space from the query point $\mathbf{x}_q$. A popular choice for $K$ is the Gaussian kernel, but other kernels may be used as well [11]. The bandwidth $\mathbf{h} \in \Re^{d \times 1}$ of the kernel is the crucial parameter that determines the local model's quality of fit. Our goal is to find a Bayesian formulation of determining $\mathbf{b}$ and $\mathbf{h}$ simultaneously.

### 2.1  Model

For the locally linear model at the query point $\mathbf{x}_q$, we can introduce hidden random variables $\mathbf{z}$ [12] and modify the linear model $y_i = \mathbf{b}^T \mathbf{x}_i$ so that $y_i = \sum_{m=1}^d z_{im} + \epsilon$, where $z_{im} = \mathbf{b}_m^T \mathbf{x}_{im} + \epsilon_{zm}$ and $\epsilon_{zm} \sim$ Normal$(0, \psi_{zm})$, $\epsilon \sim$ Normal$(0, \sigma^2)$ are both additive noise terms. Note that $\mathbf{x}_{im} = [x_{im}\ 1]^T$ and $\mathbf{b}_m = [b_m\ b_{m0}]^T$, where $x_{im}$ is the $m$th coefficient of $\mathbf{x}_i$, $b_m$ is the $m$th coefficient of $\mathbf{b}$ and $b_{m0}$ is the offset value. The $\mathbf{z}$ variables allow us to derive computationally efficient $O(d)$ EM-like updates, as we will see later. The prediction at the query point $\mathbf{x}_q$ is then $\sum_m^d \mathbf{b}_m^T \mathbf{x}_{qm}$. We assume the following prior distributions for our model, shown graphically in Fig. 1:

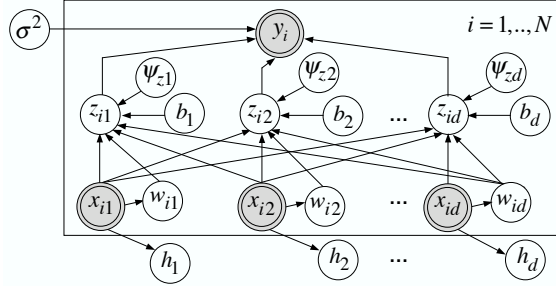

Figure 1: Graphical model. Random variables are in circles, and observed random variables are in shaded double circles.

$$p(y_i|\mathbf{z}_i) \sim \text{Normal}\left(\mathbf{1}^T \mathbf{z}_i, \sigma^2\right) \qquad p(\mathbf{b}_m|\psi_{zm}) \sim \text{Normal}\left(0, \psi_{zm} \mathbf{\Sigma}_{\mathbf{b}_m,0}\right)$$
$$p(z_{im}|\mathbf{x}_{im}) \sim \text{Normal}\left(\mathbf{b}_m^T \mathbf{x}_{im}, \psi_{zm}\right) \qquad p(\psi_{zm}) \sim \text{Scaled-Inv-}\chi^2\left(n_{m0}, \psi_{zm,0}\right)$$

where $\mathbf{1}$ is a vector of 1s, $\mathbf{z}_i \in \Re^{d \times 1}$, $z_{im}$ is the $m$th coefficient of $\mathbf{z}_i$, and $\mathbf{\Sigma}_{\mathbf{b}_m,0}$ is the prior covariance matrix of $\mathbf{b}_m$ and a $2 \times 2$ diagonal matrix. $n_{m0}$ and $\sigma^2_{mN0}$ are the prior parameters of the Scaled-inverse-$\chi^2$ distribution ($n_{m0}$ is the number of degrees of freedom parameter and $\sigma^2_{mN0}$ is the scale parameter). The Scaled-Inverse-$\chi^2$ distribution was used for $\psi_{zm}$ since it is the conjugate prior for the variance parameter of a Gaussian distribution.

In contrast to classical treatments of Bayesian weighted regression [13] where the weights enter as a heteroscedastic correction on the noise variance of each data sample, we **associate a scalar indicator-like weight,** $w_i \in \{0,1\}$**, with each sample** $\{\mathbf{x}_i, y_i\}$ in $D$. The sample is fully included in the local model if $w_i = 1$ and excluded if $w_i = 0$. We define the weight $w_i$ to be $w_i = \prod_{m=1}^d w_{im}$, where $w_{im}$ is the weight component in the $m$th input dimension. While previous methods model the weighting kernel $K$ as some explicit function, we model the weights $w_{im}$ as Bernoulli-distributed random variables, i.e., $p(w_{im}) \sim$ Bernoulli$(q_{im})$, choosing a symmetric bell-shaped function for the parameter $q_{im}$: $q_{im} = 1/(1 + (x_{im} - x_{qm})^{2r} h_m)$. $x_{qm}$ is the $m$th coefficient of $\mathbf{x}_q$, $h_m$ is the $m$th

coefficient of $\mathbf{h}$, and $r > 0$ is a positive integer[1]. As pointed out in [11], the particular mathematical formulation of a weighting kernel is largely computationally irrelevant for locally weighted learning. Our choice of function for $q_{im}$ was dominated by the desire to obtain analytically tractable learning updates. We place a Gamma prior over the bandwidth $h_m$, i.e., $p(h_m) \sim \mathrm{Gamma}\,(a_{hm0}, b_{hm0})$ where $a_{hm0}$ and $b_{hm0}$ are parameters of the Gamma distribution, to ensure that a positive weighting kernel width.

## 2.2 Inference

We can treat the entire regression problem as an EM learning problem [14, 15] and maximize the log likelihood $\log p(\mathbf{y}|\mathbf{X})$ for generating the observed data. We can maximize this incomplete log likelihood by maximizing the expected value of the complete log likelihood $p(\mathbf{y}, \mathbf{Z}, \mathbf{b}, \mathbf{w}, \mathbf{h}, \sigma^2, \psi_z|\mathbf{X}) = \prod_{i=1}^{N} p(y_i, \mathbf{z}_i, \mathbf{b}, w_i, \mathbf{h}, \sigma^2, \psi_z|\mathbf{x}_i)$. In our model, each data sample $i$ has an indicator-like scalar weight $w_i$ associated with it, allowing us to express the complete log likelihood $L$, in a similar fashion to mixture models, as:

$$
L = \log \left[ \prod_{i=1}^{N} \left[ \left[ p(y_i|\mathbf{z}_i, \sigma^2)p(\mathbf{z}_i|\mathbf{x}_i, \mathbf{b}, \psi_z) \right]^{w_i} \prod_{m=1}^{d} p(w_{im}) \right] \prod_{m=1}^{d} p(\mathbf{b}_m|\psi_{zm})p(\psi_{zm})p(h_m)p(\sigma^2) \right]
$$

Expanding the $\log p(w_{im})$ term from the expression above results in a problematic $-\log(1 + (x_{im} - x_{qm})^{2r})$ term that prevents us from deriving an analytically tractable expression for the posterior of $h_m$. To address this, we use a variational approach on concave/convex functions suggested by [16] to produce analytically tractable expressions. We can find a lower bound on the term so that $-\log(1 + (x_{im} - x_{qm})^{2r}) \geq -\lambda_{im}(x_{im} - x_{qm})^{2r} h_m$, where $\lambda_{im}$ is a variational parameter to be optimized in the M-step of our final EM-like algorithm. Our choice of weighting kernel allows us to find a lower bound to $L$ in this manner. We explored the use of other weighting kernels (e.g., a quadratic negative exponential), but had issues with finding a lower bound to the problematic terms in $\log p(w_{im})$ such that analytically tractable inference for $h_m$ could be done. The resulting lower bound to $L$ is $\hat{L}$; due to lack of space, we give the expression for $\hat{L}$ in the appendix. The expectation of $\hat{L}$ should be taken with respect to the true posterior distribution of all hidden variables $Q(\mathbf{b}, \psi_z, \mathbf{z}, \mathbf{h})$. Since this is an analytically tractable expression, a lower bound can be formulated using a technique from variational calculus where we make a factorial approximation of the true posterior, e.g., $Q(\mathbf{b}, \psi_z, \mathbf{z}, \mathbf{h}) = Q(\mathbf{b}, \psi_z)Q(\mathbf{h})Q(\mathbf{z})$ [15], that allows resulting posterior distributions over hidden variables to become analytically tractable. The posterior of $w_{im}$, $p(w_{im} = 1|y_i, \mathbf{z}_i, \mathbf{x}_i, \boldsymbol{\theta}, w_{i,k\neq m})$, is inferred using Bayes' rule:

$$
\frac{p(y_i, \mathbf{z}_i|\mathbf{x}_i, \boldsymbol{\theta}, w_{i,k\neq m}, w_{im} = 1)^{\prod_{t=1, t\neq m}^{d} \langle w_{it} \rangle} p(w_{im} = 1)}{p(y_i, \mathbf{z}_i|\mathbf{x}_i, \boldsymbol{\theta}, w_{i,k\neq m}, w_{im} = 1)^{\prod_{t=1, t\neq m}^{d} \langle w_{it} \rangle} p(w_{im} = 1) + p(w_{im} = 0)}
\tag{1}
$$

where $\boldsymbol{\theta} = \{\mathbf{b}, \psi_z, \mathbf{h}\}$ and $w_{i,k\neq m}$ denotes the set of weights $\{w_{ik}\}_{k=1, k\neq m}^{d}$. For the dimension $m$, we account for the effect of weights in the other $d - 1$ dimensions. This is a result of $w_i$ being defined as the product of weights in all dimensions. The posterior mean of $w_{im}$ is then $\langle p(w_{im} = 1|y_i, \mathbf{z}_i, \mathbf{x}_i, \boldsymbol{\theta}, w_{i,k\neq m}) \rangle$, and $\langle w_i \rangle = \prod_{m=1}^{d} \langle w_{im} \rangle$, where $\langle . \rangle$ denotes the expectation operator. We omit the full set of posterior EM update equations (please refer to the appendix for this) and list only the posterior updates for $h_m$, $w_{im}$, $\mathbf{b}_m$ and $\mathbf{z}_i$:

$$
\boldsymbol{\Sigma}_{\mathbf{b}_m} = \left( \boldsymbol{\Sigma}_{\mathbf{b}_m,0}^{-1} + \sum_{i=1}^{N} \langle w_i \rangle \mathbf{x}_{im}\mathbf{x}_{im}^{T} \right)^{-1} \qquad \boldsymbol{\Sigma}_{\mathbf{z}_i|y_i, \mathbf{x}_i} = \frac{\boldsymbol{\Psi}_{zN}}{\langle w_i \rangle} - \frac{1}{s_i} \left( \frac{\boldsymbol{\Psi}_{zN}}{\langle w_i \rangle} \mathbf{1}\mathbf{1}^{T} \frac{\boldsymbol{\Psi}_{zN}}{\langle w_i \rangle} \right)
$$

$$
\langle \mathbf{b}_m \rangle = \boldsymbol{\Sigma}_{\mathbf{b}_m} \left( \sum_{i=1}^{N} \langle w_i \rangle \langle \mathbf{z}_{im} \rangle \mathbf{x}_{im} \right) \qquad \langle \mathbf{z}_i \rangle = \frac{\boldsymbol{\Psi}_{zN}\mathbf{1}}{s_i \langle w_i \rangle} + \left( \mathbf{I}_{d,d} - \frac{\boldsymbol{\Psi}_{zN}\mathbf{1}\mathbf{1}^{T}}{s_i \langle w_i \rangle} \right) \mathbf{b}\mathbf{x_i}
$$

$$
\langle w_{im} \rangle = \frac{q_{im} A_i^{\prod_{k=1, k\neq m}^{d} \langle w_{ik} \rangle}}{q_{im} A_i^{\prod_{k=1, k\neq m}^{d} \langle w_{ik} \rangle} + 1 - q_{im}} \qquad \langle h_m \rangle = \frac{a_{hm0} + N - \sum_{i=1}^{N} \langle w_{im} \rangle}{b_{hm0} + \sum_{i=1}^{N} \lambda_{im}(x_{im} - x_{qm})^{2r}}
$$

where $\mathbf{I}_{d,d}$ is a $d \times d$ identity matrix, $\mathbf{bx_i}$ is a $d$ by 1 vector with coefficients $\langle \mathbf{b}_m \rangle^T \mathbf{x}_{im}$, $\langle w_i \rangle = \prod_{m=1}^d \langle w_{im} \rangle$, $\boldsymbol{\Psi}_{zN}$ is a diagonal matrix with $\psi_{zN}$ on its diagonal, $s_i = \sigma^2 + \mathbf{1}^T \frac{\boldsymbol{\Psi}_{zN}}{\langle w_i \rangle} \mathbf{1}$ (to avoid division by zero, $\langle w_i \rangle$ needs to be capped to some small non-zero value), $q_{im} = \lambda_{im} = 1/(1+(x_{im} - x_{qm})^{2r} \langle h_m \rangle)$, and $A_i = N(y_i; \mathbf{1}^T \langle \mathbf{z}_i \rangle, \sigma^2) \prod_{m=1}^d N(z_{im}; \langle \mathbf{b}_m \rangle^T \mathbf{x}_{im}, \psi_{zm})$. Closer examination of the expression for $\langle \mathbf{b}_m \rangle$ shows that it is a standard Bayesian weighted regression update [13], i.e., a data sample $i$ with lower weight $w_i$ will be downweighted in the regression. Since the weights are influenced by the residual error at each data point (see posterior update for $\langle w_{im} \rangle$), an outlier will be downweighted appropriately and eliminated from the local model. Fig. 2 shows how local kernel shaping is able to ignore outliers that a classical GP fits.

A few remarks should be made regarding the initialization of priors used in the posterior EM updates. $\boldsymbol{\Sigma}_{\mathbf{b}_m,0}$ can be set to $10^6 \mathbf{I}$ to reflect a large uncertainty associated with the prior distribution of $\mathbf{b}$. The initial noise variance, $\psi_{zm,0}$, should be set to the best guess on the noise variance. To adjust the strength of this prior, $n_{m0}$ can be set to the number of samples one believes to have seen with noise variance $\psi_{zm,0}$ Finally, the initial $h$ of the weighting kernel should be set so that the kernel is broad and wide. We use values of $a_{hm0} = b_{hm0} = 10^{-6}$ so that $h_{m0} = 1$ with high uncertainty. Note that some sort of initial belief about the noise level is needed to distinguish between noise and structure in the training data. Aside from the initial prior on $\psi_{zm}$, we used the same priors for all data sets in our evaluations.

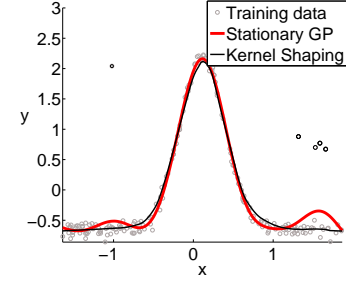

Figure 2: Effect of outliers (in black circles)

## 2.3 Computational Complexity

For one local model, the EM update equations have *a computational complexity of $O(Nd)$ per EM iteration*, where $d$ is the number input dimensions and $N$ is the size of the training set. This efficiency arises from the introduction of the hidden random variables $\mathbf{z}_i$, which allows $\langle \mathbf{z}_i \rangle$ and $\boldsymbol{\Sigma}_{\mathbf{z}_i|y_i,\mathbf{x}_i}$ to be computed in $O(d)$ and avoids a $d \times d$ matrix inversion which would typically require $O(d^3)$. Some nonstationary GP methods, e.g., [5], require $O(N^3) + O(N^2)$ for training and prediction, while other more efficient stationary GP methods, e.g., [17], require $O(M^2N) + O(M^2)$ training and prediction costs (where $M << N$ is the number of pseudoinputs used in [17]). Our algorithm requires $O(NdI_{EM})$, where $I_{EM}$ is the number of EM iterations—with a maximal cap of 1000 iterations used. Our algorithm also does not require any MCMC sampling as in [8, 9], making it more appealing to real-time applications.

## 3 Extension to Gaussian Processes

We can apply the algorithm in section 2 not only to locally weighted learning with linear models, but also to derive a nonstationary GP method. A GP is defined by a mean and and a covariance function, where the covariance function $K$ captures dependencies between any two points as a function of the corresponding inputs, i.e., $k(\mathbf{x}_i, \mathbf{x}_j) = \text{cov}(f(\mathbf{x}_i), f(\mathbf{x}'_j))$, where $i, j = 1, .., N$. Standard GP models use a stationary covariance function, where the covariance between any two points in the training data is a function of the distances $|\mathbf{x}_i - \mathbf{x}_j|$, not of their locations. Stationary GPs perform suboptimally for functions that have different properties in various parts of the input space (e.g., discontinuous functions) where the stationary assumption fails to hold. Various methods have been proposed to specify nonstationary GPs. These include defining a nonstationary Matérn covariance function [5], adopting a mixture of local experts approach [18, 8, 9] to use independent GPs to cover data in different regions of the input space, and using multidimensional scaling to map a nonstationary spatial GP into a latent space [19].

Given the data set $D$ drawn from the function $y = f(\mathbf{x})+\epsilon$, as previously introduced in section 2, we propose an approach to specify a nonstationary covariance function. Assuming the use of a quadratic negative exponential covariance function, the covariance function of a stationary GP is $k(\mathbf{x}_i, \mathbf{x}_j) = v_1^2 \exp(-0.5 \sum_{m=1}^d h_m(x_{im} - x'_{jm})^2) + v_0$, where the hyperparameters $\{h_1, h_2, ..., h_d, v_0, v_1\}$ are

optimized. In a nonstationary GP, the covariance function could then take the form[2] $k(\mathbf{x}_i, \mathbf{x}_j) = v_1^2 \exp\left(-0.5 \sum_{m=1}^{d} (x_{im} - x_{jm})^2 \frac{h_{im} h_{jm}}{(h_{im} + h_{jm})}\right) + v_0$, where $h_{im}$ is the bandwidth of the local model centered at $x_{im}$ and $h_{jm}$ is the bandwidth of the local model centered at $x_{jm}$. We learn first the values of $\{h_{im}\}_{m=1}^{d}$ for all training data samples $i = 1, ..., N$ using our proposed local kernel shaping algorithm and then optimize the hyperparameters $v_0$ and $v_1$. To make a prediction for a test sample $\mathbf{x}_q$, we learn also the values of $\{h_{qm}\}_{m=1}^{d}$, i.e., the bandwidth of the local model centered at $\mathbf{x}_q$. Importantly, since the covariance function of the GP is derived from locally constant models, we learn with locally constant, instead of locally linear, polynomials. We use $r = 1$ for the weighting kernel in order keep the degree of nonlinearity consistent with that in the covariance function (i.e., quadratic). Even though the weighting kernel used in the local kernel shaping algorithm is not a quadratic negative exponential, it has a similar bell shape, but with a flatter top and shorter tails. Because of this, our augmented GP is an approximated form of a nonstationary GP. Nonetheless, it is able to capture nonstationary properties of the function $f$ without needing MCMC sampling, unlike previously proposed nonstationary GP methods [8, 9].

## 4 Experimental Results

### 4.1 Synthetic Data

First, we show our local kernel shaping algorithm's bandwidth adaptation abilities on several synthetic data sets, comparing it to a stationary GP and our proposed augmented nonstationary GP. For the ease of visualization, we consider the following one-dimensional functions, similar to those in [5]: i) a function with a discontinuity, ii) a spatially inhomogeneous function, and iii) a straight line function. The data set for function i) consists of 250 training samples, 201 test inputs (evenly spaced across the input space) and output noise with $\sigma^2 = 0.3025$; the data set for function ii) consists of 250 training samples, 101 test inputs and an output signal-to-noise ratio (SNR) of 10; and the data set for function iii) has 50 training samples, 21 test inputs and an output SNR of 100.

Fig. 3 shows the predicted outputs of a stationary GP, augmented nonstationary GP and the local kernel shaping algorithm for data sets i)-iii). The local kernel shaping algorithm smoothes over regions where a stationary GP overfits and yet, it still manages to capture regions of highly varying curvature, as seen in Figs. 3(a) and 3(b). It correctly adjusts the bandwidths $h$ with the curvature of the function. When the data looks linear, the algorithm opens up the weighting kernel so that all data samples are considered, as Fig. 3(c) shows. Our proposed augmented nonstationary GP also can handle the nonstationary nature of the data sets as well, and its performance is quantified in Table 1. Returning to our motivation to use these algorithms to obtain linearizations for learning control, it is important to realize that the high variations from fitting noise, as shown by the stationary GP in Fig. 3, are detrimental for learning algorithms, as the slope (or tangent hyperplane, for high-dimensional data) would be wrong.

Table 1 reports the normalized mean squared prediction error (nMSE) values for function i) and function ii) data sets, averaged over 20 random data sets. Fig. 4 shows results of the local kernel shaping algorithm and the proposed augmented nonstationary GP on the "real-world" motorcycle data set [20] consisting of 133 samples (with 80 equally spaced input query points used for prediction). We also show results from a previously proposed MCMC-based nonstationary GP method: an alternate infinite mixture of GP experts [9]. We can see that the augmented nonstationary GP and the local kernel shaping algorithm both capture the leftmost flatter region of the function, as well as some of the more nonlinear and noisier regions after 30msec.

### 4.2 Robot Data

Next, we move on to an example application: learning an inverse kinematics model for a 3 degree-of-freedom (DOF) haptic robot arm (manufactured by SensAble, shown in Fig. 5(a)) in order to control the end-effector along a desired trajectory. This will allow us to verify that the kernel shaping algo-

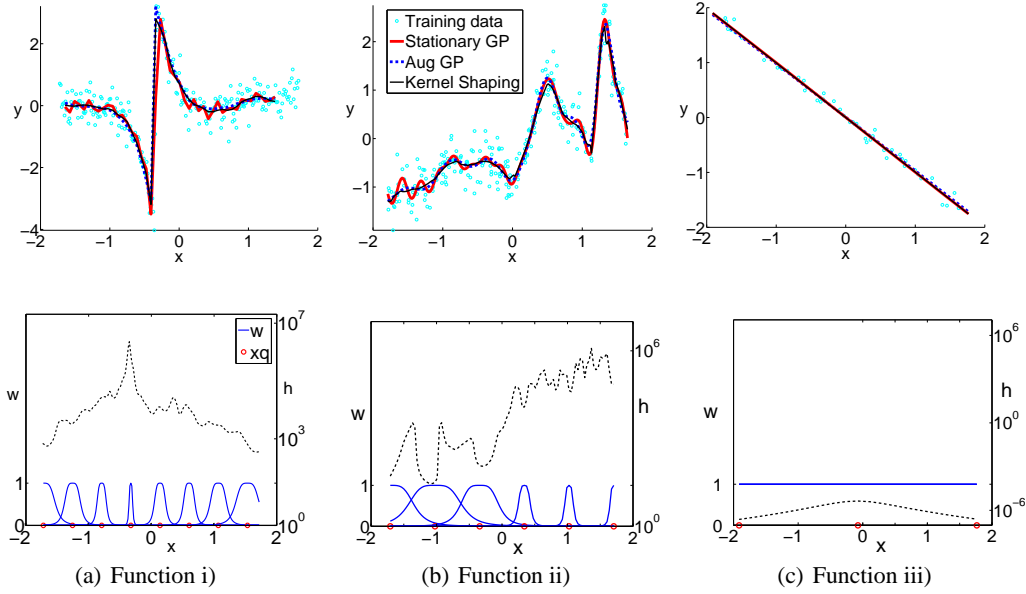

(a) Function i)             (b) Function ii)             (c) Function iii)

Figure 3: Predicted outputs using a stationary GP, our augmented nonstationary GP and local kernel shaping. Figures on the bottom show the bandwidths learnt by local kernel shaping and the corresponding weighting kernels (in dotted black lines) for input query points (shown in red circles).

rithm can successfully deal with a large, noisy real-world data set with outliers and non-stationary properties—typical characteristics of most control learning problems.

We collected $60,000$ data samples from the arm while it performed random sinusoidal movements within a constrained box volume of Cartesian space. Each sample consists of the arm's joint angles $\mathbf{q}$, joint velocities $\dot{\mathbf{q}}$, end-effector position in Cartesian space $\mathbf{x}$, and end-effector velocities $\dot{\mathbf{x}}$. From this data, we first learn a forward kinematics model: $\dot{\mathbf{x}} = \mathbf{J}(\mathbf{q})\dot{\mathbf{q}}$, where $\mathbf{J}(\mathbf{q})$ is the Jacobian matrix. The transformation from $\dot{\mathbf{q}}$ to $\dot{\mathbf{x}}$ can be assumed to be locally linear at a particular configuration $\mathbf{q}$ of the robot arm. We learn the forward model using kernel shaping, building a local model around each training point only if that point is not already sufficiently covered by an existing local model (e.g., having an activation weight of less than 0.2). Using insights into robot geometry, we localize the models only with respect to $\mathbf{q}$ while the regression of each model is trained only on a mapping from $\dot{\mathbf{q}}$ to $\dot{\mathbf{x}}$—these geometric insights are easily incorporated as priors in the Bayesian model. This procedure resulted in 56 models being built to cover the entire space of training data.

We artificially introduce a redundancy in our inverse kinematics problem on the 3-DOF arm by specifying the desired trajectory $(\mathbf{x}, \dot{\mathbf{x}})$ only in terms of $x$, $z$ positions and velocities, i.e., the movement is supposed to be in a vertical plane in front of the robot. Analytically, the inverse kinematics equation is $\dot{\mathbf{q}} = \mathbf{J}^{\#}(\mathbf{q})\dot{\mathbf{x}} - \alpha(\mathbf{I} - \mathbf{J}^{\#}\mathbf{J})\frac{\partial g}{\partial \mathbf{q}}$, where $J^{\#}(\mathbf{q})$ is the pseudo-inverse of the Jacobian. The second term is an optimal solution to the redundancy problem, specified here by a cost function $g$ in terms of joint angles $\mathbf{q}$. To learn a model for $\mathbf{J}^{\#}$, we can reuse the local regions of $\mathbf{q}$ from the forward model, where $\mathbf{J}^{\#}$ is also locally linear. The redundancy issue can be solved by applying an additional weight to each data point according to a reward function [21]. In our case, the task is specified in terms of $\{\dot{x}, \dot{z}\}$, so we define a reward based on a desired $y$ coordinate, $y_{des}$, that we would like to enforce as a soft constraint. Our reward function is $g = e^{-\frac{1}{2}h(k(y_{des}-y)-\dot{y})^2}$, where $k$ is a gain and $h$ specifies the steepness of the reward. This ensures that the learnt inverse model chooses a solution which produces a $\dot{y}$ that pushes the $y$ coordinate toward $y_{des}$. We invert each forward local model using a weighted linear regression, where each data point is weighted by the weight from the forward model and additionally weighted by the reward.

We test the performance of this inverse model (Learnt IK) in a figure-eight tracking task as shown in Fig. 5(b). As seen, the learnt model performs as well as the analytical inverse kinematics solution (IK), with root mean squared tracking errors in positions and velocities very close to that of the

Table 1: Average normalized mean squared prediction error values, for a stationary GP model, our augmented nonstationary GP, local kernel shaping—averaged over 20 random data sets.

| Method | Function i) | Function ii) |
|---|---|---|
| Stationary GP | $0.1251 \pm 0.013$ | $0.0230 \pm 0.0047$ |
| Augmented nonstationary GP | $0.0110 \pm 0.0078$ | $0.0212 \pm 0.0067$ |
| Local Kernel Shaping | $0.0092 \pm 0.0068$ | $0.0217 \pm 0.0058$ |

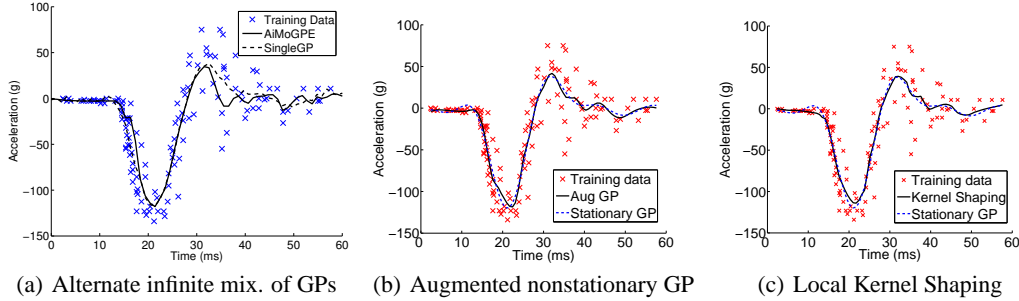

(a) Alternate infinite mix. of GPs    (b) Augmented nonstationary GP    (c) Local Kernel Shaping

Figure 4: Motorcycle impact data set from [20], with predicted results shown for our augmented GP and local kernel shaping algorithms. Results from the alternate infinite mixture of GP experts (AiMoGPE) are taken from [9].

analytical solution. This demonstrates that kernel shaping is an effective learning algorithm for use in robot control learning applications.

Applying any arbitrary nonlinear regression method (such as a GP) to the inverse kinematics problem would, in fact, lead to unpredictably bad performance. The inverse kinematics problem is a one-to-many mapping and requires careful design of a learning problem to avoid problems with non-convex solution spaces [22]. Our suggested method of learning linearizations with a forward mapping (which is a proper function), followed by learning an inverse mapping within the local region of the forward mapping, is one of the few clean approaches to the problem. Instead of using locally linear methods, one could also use density-based estimation techniques like mixture models [23]. However, these methods must select the correct mode in order to arrive at a valid solution, and this final step may be computationally intensive or involve heuristics. For these reasons, applying a MCMC-type approach or GP-based method to the inverse kinematics problem was omitted as a comparison.

## 5 Discussion

We presented a full Bayesian treatment of nonparametric local multi-dimensional kernel adaptation that simultaneously estimates the regression and kernel parameters. The algorithm can also be integrated into nonlinear algorithms, offering a valuable and flexible tool for learning. We show that our local kernel shaping method is particularly useful for learning control, demonstrating results on an inverse kinematics problem, and envision extensions to more complex problems with redundancy,

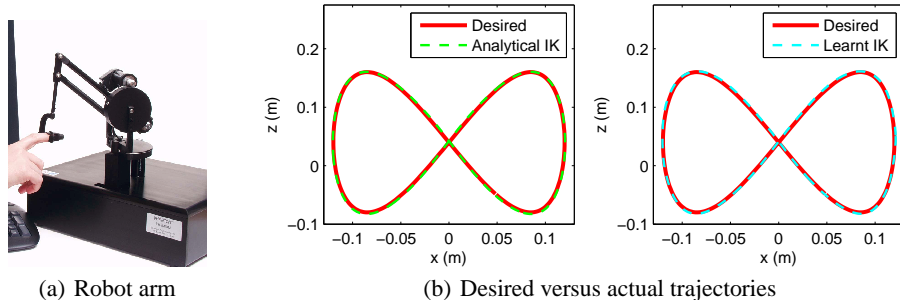

(a) Robot arm    (b) Desired versus actual trajectories

Figure 5: Desired versus actual trajectories for SensAble Phantom robot arm

e.g., learning inverse dynamics models of complete humanoid robots. Note that our algorithm requires only one prior be set by the user, i.e., the prior on the output noise. All other biases are initialized the same for all data sets and kept uninformative. In its current form, our Bayesian kernel shaping algorithm is built for high-dimensional inputs due to its low computational complexity— it scales linearly with the number of input dimensions. However, numerical problems may arise in case of redundant and irrelevant input dimensions. Future work will address this issue through the use of an automatic relevant determination feature. Other future extensions include an online implementation of the local kernel shaping algorithm.

## Footnotes

[1]$(x_{im} - x_{qm})$ is taken to the power $2r$ in order to ensure that the resulting expression is positive. Adjusting $r$ affects how long the tails of the kernel are. We use $r = 2$ for all our experiments.

[2]This is derived from the definition of $K$ as a positive semi-definite matrix, i.e. where the integral is the product of two quadratic negative exponentials—one with parameter $h_{im}$ and the other with parameter $h_{jm}$.

## References

[1] C. K. I. Williams and C. E. Rasmussen. Gaussian processes for regression. In David S. Touretzky, Michael C. Mozer, and Michael E. Hasselmo, editors, *In Advances in Neural Information Processing Systems 8*, volume 8. MIT Press, 1995.

[2] J. H. Friedman. A variable span smoother. Technical report, Stanford University, 1984.

[3] T. Poggio and F. Girosi. Regularization algorithms for learning that are equivalent to multilayer networks. *Science*, 247:213–225, 1990.

[4] J. Fan and I. Gijbels. *Local polynomial modeling and its applications*. Chapman and Hall, 1996.

[5] C. J. Paciorek and M. J. Schervish. Nonstationary covariance functions for Gaussian process regression. In *Advances in Neural Information Processing Systems 16*. MIT Press, 2004.

[6] J. Fan and I. Gijbels. Data-driven bandwidth selection in local polynomial fitting: Variable bandwidth and spatial adaptation. *Journal of the Royal Statistical Society B*, 57:371–395, 1995.

[7] S. Schaal and C.G. Atkeson. Assessing the quality of learned local models. In G. Tesauro J. Cowan and J. Alspector, editors, *Advances in Neural Information Processing Systems*, pages 160–167. Morgan Kaufmann, 1994.

[8] C. E. Rasmussen and Z. Ghahramani. Infinite mixtures of Gaussian processes. In *Advances in Neural Information Processing Systems 14*. MIT Press, 2002.

[9] E. Meeds and S. Osindero. An alternative infinite mixture of Gaussian process experts. In *Advances in Neural Information Processing Systems 17*. MIT Press, 2005.

[10] C. Atkeson and S. Schaal. Robot learning from demonstration. In *Proceedings of the 14th international conference on Machine learning*, pages 12–20. Morgan Kaufmann, 1997.

[11] C. Atkeson, A. Moore, and S. Schaal. Locally weighted learning. *AI Review*, 11:11–73, April 1997.

[12] A. D'Souza, S. Vijayakumar, and S. Schaal. The Bayesian backfitting relevance vector machine. In *Proceedings of the 21st International Conference on Machine Learning*. ACM Press, 2004.

[13] A. Gelman, J. Carlin, H.S. Stern, and D.B. Rubin. *Bayesian Data Analysis*. Chapman and Hall, 2000.

[14] A. Dempster, N. Laird, and D. Rubin. Maximum likelihood from incomplete data via the EM algorithm. *Journal of Royal Statistical Society. Series B*, 39(1):1–38, 1977.

[15] Z. Ghahramani and M.J. Beal. Graphical models and variational methods. In D. Saad and M. Opper, editors, *Advanced Mean Field Methods - Theory and Practice*. MIT Press, 2000.

[16] T. S. Jaakkola and M. I. Jordan. Bayesian parameter estimation via variational methods. *Statistics and Computing*, 10:25–37, 2000.

[17] E. Snelson and Z. Ghahramani. Sparse Gaussian processes using pseudo-inputs. In *Advances in Neural Information Processing Systems 18*. MIT Press, 2006.

[18] V. Tresp. Mixtures of Gaussian processes. In *Advances in Neural Information Processing Systems 13*. MIT Press, 2000.

[19] A. M. Schmidt and A. O'Hagan. Bayesian inference for nonstationary spatial covariance structure via spatial deformations. *Journal of Royal Statistical Society. Series B*, 65:745–758, 2003.

[20] B. W. Silverman. Some aspects of the spline smoothing approach to non-parametric regression curve fitting. *Journal of Royal Statistical Society. Series B*, 47:1–52, 1985.

[21] J. Peters and S. Schaal. Learning to control in operational space. *International Journal of Robotics Research*, 27:197–212, 2008.

[22] M. I. Jordan and D. E. Rumelhart. Internal world models and supervised learning. In *Machine Learning: Proceedings of Eighth Internatinoal Workshop*, pages 70–85. Morgan Kaufmann, 1991.

[23] Z. Ghahramani. Solving inverse problems using an EM approach to density estimation. In *Proceedings of the 1993 Connectionist Models summer school*, pages 316–323. Erlbaum Associates, 1994.
